# The Convergence of Contrastive Divergences

**Alan Yuille**
Department of Statistics
University of California at Los Angeles
Los Angeles, CA 90095
`yuille@stat.ucla.edu`

## Abstract

This paper analyses the Contrastive Divergence algorithm for learning statistical parameters. We relate the algorithm to the stochastic approximation literature. This enables us to specify conditions under which the algorithm is guaranteed to converge to the optimal solution (with probability 1). This includes necessary and sufficient conditions for the solution to be unbiased.

## 1 Introduction

Many learning problems can be reduced to statistical inference of parameters. But inference algorithms for this task tend to be very slow. Recently Hinton proposed a new algorithm called contrastive divergences (CD) [1]. Computer simulations show that this algorithm tends to converge, and to converge rapidly, although not always to the correct solution [2]. Theoretical analysis shows that CD can fail but does not give conditions which guarantee convergence [3,4].

This paper relates CD to the stochastic approximation literature [5,6] and hence derives elementary conditions which ensure convergence (with probability 1). We conjecture that far stronger results can be obtained by applying more advanced techniques such as those described by Younes [7]. We also give necessary and sufficient conditions for the solution of CD to be unbiased.

Section (2) describes CD and shows that it is closely related to a class of stochastic approximation algorithms for which convergence results exist. In section (3) we state and give a proof of a simple convergence theorem for stochastic approximation algorithms. Section (4) applies the theorem to give sufficient conditions for convergence of CD.

## 2 Contrastive Divergence and its Relations

The task of statistical inference is to estimate the model parameters $\omega^*$ which minimize the Kullback-Leibler divergence $D(P_0(x)||P(x|\omega))$ between the empirical distribution func-

tion of the observed data $P_0(x)$ and the model $P(x|\omega)$. It is assumed that the model distribution is of the form $P(x|\omega) = e^{-E(x;\omega)}/Z(\omega)$.

Estimating the model parameters is difficult. For example, it is natural to try performing steepest descent on $D(P_0(x)||P(x|\omega))$. The *steepest descent algorithm* can be expressed as:

$$\omega_{t+1} - \omega_t = \gamma_t \{-\sum_x P_0(x)\frac{\partial E(x;\omega)}{\partial \omega} + \sum_x P(x|\omega)\frac{\partial E(x;\omega)}{\partial \omega}\}, \qquad (1)$$

where the $\{\gamma_t\}$ are constants.

Unfortunately steepest descent is usually computationally intractable because of the need to compute the second term on the right hand side of equation (1). This is extremely difficult because of the need to evaluate the normalization term $Z(\omega)$ of $P(x|\omega)$.

Moreover, steepest descent also risks getting stuck in a local minimum. There is, however, an important exception if we can express $E(x;\omega)$ in the special form $E(x;\omega) = \omega \cdot \phi(x)$, for some function $\phi(x)$. In this case $D(P_0(x)||P(x|\omega))$ is convex and so steepest descent is guaranteed to converge to the global minimum. But the difficulty of evaluating $Z(\omega)$ remains.

The CD algorithm is formally similar to steepest descent. But it avoids the need to evaluate $Z(\omega)$. Instead it approximates the second term on the right hand side of the steepest descent equation (1) by a stochastic term. This approximation is done by defining, for each $\omega$, a Markov Chain Monte Carlo (MCMC) transition kernel $K_\omega(x, y)$ whose invariant distribution is $P(x|\omega)$ (i.e. $\sum_x P(x|\omega)K_\omega(x, y) = P(y|\omega)$).

Then the *CD algorithm* can be expressed as:

$$\omega_{t+1} - \omega_t = \gamma_t \{-\sum_x P_0(x)\frac{\partial E(x;\omega)}{\partial \omega} + \sum_x Q_\omega(x)\frac{\partial E(x;\omega)}{\partial \omega}\}, \qquad (2)$$

where $Q_\omega(x)$ is the empirical distribution function on the samples obtained by initializing the chain at the data samples $P_0(x)$ and running the Markov chain forward for $m$ steps (the value of $m$ is a design choice).

We now observe that CD is similar to a class of *stochastic approximation algorithms* which also use MCMC methods to stochastically approximate the second term on the right hand side of the steepest descent equation (1). These algorithms are reviewed in [7] and have been used, for example, to learn probability distributions for modelling image texture [8].

A typical algorithm of this type introduces a state vector $S^t(x)$ which is initialized by setting $S^{t=0}(x) = P_0(x)$. Then $S^t(x)$ and $\omega_t$ are updated sequentially as follows. $S^t(x)$ is obtained by sampling with the transition kernel $K_{\omega_t}(x, y)$ using $S^{t-1}(x)$ as the initial state for the chain. Then $\omega_{t+1}$ is computed by replacing the second term in equation (1) by the expectation with respect to $S^t(x)$. From this perspective, we can obtain CD by having a state vector $S^t(x) (= Q_\omega(x))$ which gets re-initialized to $P_0(x)$ at each time step.

This stochastic approximation algorithm, and its many variants, have been extensively studied and convergence results have been obtained (see [7]). The convergence results are based on stochastic approximation theorems [6] whose history starts with the analysis of the Robbins-Monro algorithm [5]. Precise conditions can be specified which guarantee convergence in probability. In particular, Kushner [9] has proven convergence to global optima. Within the NIPS community, Orr and Leen [10] have studied the ability of these algorithms to escape from local minima by basin hopping.

# 3 Stochastic Approximation Algorithms and Convergence

The general stochastic approximation algorithm is of the form:

$$\omega_{t+1} = \omega_t - \gamma_t S(\omega_t, N_t),  \tag{3}$$

where $N_t$ is a random variable sampled from a distribution $P_n(N)$, $\gamma_t$ is the damping coefficient, and $S(.,.)$ is an arbitrary function.

We now state a theorem which gives sufficient conditions to ensure that the stochastic approximation algorithm (3) converges to a (solution) state $\omega^*$. The theorem is chosen because of the simplicity of its proof and we point out that a large variety of alternative results are available, see [6,7,9] and the references they cite.

The theorem involves three basic concepts. The first is a function $L(\omega) = (1/2)|\omega - \omega^*|^2$ which is a measure of the distance of the current state $\omega$ from the solution state $\omega^*$ (in the next section we will require $\omega^* = \arg\min_\omega D(P_0(x)||P(x|\omega)))$. The second is the expected value $\sum_N P_n(N)S(\omega, N)$ of the update term in the stochastic approximation algorithm (3). The third is the expected squared magnitude $\langle|S(\omega, N)|^2\rangle$ of the update term.

The theorem states that the algorithm will converge provided three conditions are satisfied. These conditions are fairly intuitive. The first condition requires that the expected update $\sum_N P_n(N)S(\omega, N)$ has a large component towards the solution $\omega^*$ (i.e. in the direction of the negative gradient of $L(\omega)$). The second condition requires that the expected squared magnitude $\langle|S(\omega, N)|^2\rangle$ is bounded, so that the "noise" in the update is not too large. The third condition requires that the damping coefficients $\gamma_t$ decrease with time $t$, so that the algorithm eventually settles down into a fixed state. This condition is satisfied by setting $\gamma_t = 1/t$, $\forall t$ (which is the fastest fall off rate consistent with the SAC theorem).

We now state the theorem and briefly sketch the proof which is based on martingale theory (for an introduction, see [11]).

**Stochastic Approximation Convergence (SAC) Theorem**. *Consider the stochastic approximation algorithm, equation (3), and let $L(\omega) = (1/2)|\omega - \omega^*|^2$. Then the algorithm will converge to $\omega^*$ with probability 1 provided: (1) $-\nabla L(\omega) \cdot \sum_N P_n(N)S(\omega, N) \geq K_1 L(\omega)$ for some constant $K_1$, (2) $\langle|S(\omega, N)|^2\rangle_t \leq K_2(1 + L(\omega))$, where $K_2$ is some constant and the expectation $\langle.\rangle_t$ is taken with respect to all the data prior to time $t$, and (3) $\sum_{t=1}^\infty \gamma_t = \infty$ and $\sum_{t=1}^\infty \gamma_t^2 < \infty$.*

Proof. *The proof [12] is a consequence of the supermartingale convergence theorem [11]. This theorem states that if $X_t, Y_t, Z_t$ are positive random variables obeying $\sum_{t=0}^\infty Y_t \leq \infty$ with probability one and $\langle X_{t+1}\rangle \leq X_t + Y_t - Z_t$, $\forall t$, then $X_t$ converges with probability 1 and $\sum_{t=0}^\infty Z_t < \infty$. To apply the theorem, set $X_t = (1/2)|\omega_t - \omega^*|^2$, set $Y_t = (1/2)K_2\gamma_t^2$ and $Z_t = -X_t(K_2\gamma_t^2 - K_1\gamma_t)$ ($Z_t$ is positive for sufficiently large t). Conditions 1 and 2 imply that $X_t$ can only converge to 0. The result follows after some algebra.*

# 4 CD and SAC

The CD algorithm can be expressed as a stochastic approximation algorithm by setting:

$$S(\omega_t, N_t) = -\sum_x P_0(x)\frac{\partial E(x; \omega)}{\partial \omega} + \sum_x Q_\omega(x)\frac{\partial E(x; \omega)}{\partial \omega},  \tag{4}$$

where the random variable $N_t$ corresponds to the MCMC sampling used to obtain $Q_\omega(x)$.

We can now apply the SAC to give three conditions which guarantee convergence of the CD algorithm. The third condition can be satisfied by setting $\gamma_t = 1/t$, $\forall t$. We can satisfy the second condition by requiring that the gradient of $E(x; \omega)$ with respect to $\omega$ is bounded, see equation (4). We conjecture that weaker conditions, such as requiring only that the gradient of $E(x; \omega)$ be bounded by a function linear in $\omega$, can be obtained using the more sophisticated martingale analysis described in [7].

It remains to understand the first condition and to determine whether the solution is unbiased. These require studying the *expected CD update*:

$$\sum_{N_t} P_n(N_t) S(\omega_t, N_t) = -\sum_x P_0(x) \frac{\partial E(x; \omega)}{\partial \omega} + \sum_{y,x} P_0(y) K_\omega^m(y, x) \frac{\partial E(x; \omega)}{\partial \omega}, \quad (5)$$

which is derived using the fact that the expected value of $Q_\omega(x)$ is $\sum_y P_0(y) K_\omega^m(y, x)$ (where the superscript $^m$ indicates running the transition kernel $m$ times).

We now re-express this expected CD update in two different ways, Results 1 and 2, which give alternative ways of understanding it. We then proceed to Results 3 and 4 which give conditions for convergence and unbiasedness of CD.

But we must first introduce some background material from Markov Chain theory [13].

We choose the transition kernel $K_\omega(x, y)$ to satisfy *detailed balance* so that $P(x|\omega) K_\omega(x, y) = P(y|\omega) K_\omega(y, x)$. Detailed balance is obeyed by many MCMC algorithms and, in particular, is always satisfied by Metropolis-Hasting algorithms. It implies that $P(x|\omega)$ is the invariant kernel of $K_\omega(x, y)$ so that $\sum_x P(x|\omega) K_\omega(x, y) = P(y|\omega)$ (all transition kernels satisfy $\sum_y K_\omega(x, y) = 1$, $\forall x$).

Detailed balance implies that the matrix $Q_\omega(x, y) = P(x|\omega)^{1/2} K_\omega(x, y) P(y|\omega)^{-1/2}$ is symmetric and hence has orthogonal eigenvectors and eigenvalues $\{e_\omega^\mu(x), \lambda_\omega^\mu\}$. The eigenvalues are ordered by magnitude (largest to smallest). The first eigenvalue is $\lambda^1 = 1$ (so $|\lambda^\mu| < 1$, $\mu \geq 2$). By standard linear algebra, we can write $Q_\omega(x, y)$ in terms of its eigenvectors and eigenvalues $Q_\omega(x, y) = \sum_\mu \lambda_\omega^\mu e_\omega^\mu(x) e_\omega^\mu(y)$, which implies that we can express the transition kernel applied $m$ times by:

$$K_\omega^m(x, y) = \sum_\mu \{\lambda_\omega^\mu\}^m \{P(x|\omega)\}^{-1/2} e_\omega^\mu(x) \{P(y|\omega)\}^{1/2} e_\omega^\mu(y) = \sum_\mu \{\lambda_\omega^\mu\}^m u_\omega^\mu(x) v_\omega^\mu(y),$$
$$(6)$$

where the $\{v_\omega^\mu(x)\}$ and $\{u_\omega^\mu(x)\}$ are the *left and right eigenvectors* of the transition kernel $K_\omega(x, y)$. They are defined by:

$$v_\omega^\mu(x) = e_\omega^\mu(x) \{P(x|\omega)\}^{1/2}, \ u_\omega^\mu(x) = e_\omega^\mu(x) \{P(x|\omega)\}^{-1/2}, \ \forall \mu, \quad (7)$$

and it can be verified that $\sum_x v_\omega^\mu(x) K_\omega(x, y) = \lambda_\omega^\mu v_\omega^\mu(y)$, $\forall \mu$ and $\sum_y K_\omega(x, y) u_\omega^\mu(y) = \lambda_\omega^\mu u_\omega^\mu(x)$, $\forall \mu$. In addition, the left and right eigenvectors are mutually orthonormal so that $\sum_x v_\omega^\mu(x) u_\omega^\nu(x) = \delta_{\mu\nu}$, where $\delta_{\mu\nu}$ is the Kronecker delta function. This implies that we can express any function $f(x)$ in equivalent expansions,

$$f(x) = \sum_\mu \{\sum_y f(y) u_\omega^\mu(y)\} v_\omega^\mu(x), \ \ f(x) = \sum_\mu \{\sum_y f(y) v_\omega^\mu(y)\} u_\omega^\mu(x). \quad (8)$$

Moreover, the first left and right eigenvectors can be calculated explicitly to give:

$$v_\omega^1(x) = P(x|\omega), \ u_\omega^1(x) \propto 1, \ \lambda_\omega^1 = 1, \tag{9}$$

which follows because $P(x|\omega)$ is the (unique) invariant distribution of the transition kernel $K_\omega(x, y)$ and hence is the first left eigenvector.

We now have sufficient background to state and prove our first result.

**Result 1**. *The expected CD update corresponds to replacing the update term $\sum_x P(x|\omega) \frac{\partial E(x;\omega)}{\partial \omega}$ in the steepest descent equation (1) by:*

$$\sum_x \frac{\partial E(x;\omega)}{\partial \omega} P(x|\omega) + \sum_{\mu=2} \{\lambda_\omega^\mu\}^m \{\sum_y P_0(y) u_\omega^\mu(y)\} \{\sum_x v_\omega^\mu(x) \frac{\partial E(x;\omega)}{\partial \omega}\}, \tag{10}$$

*where $\{v_\omega^\mu(x), u_\omega^\mu(x)\}$ are the left and right eigenvectors of $K_\omega(x, y)$ with eigenvalues $\{\lambda^\mu\}$.*

Proof.
*The expected CD update replaces $\sum_x P(x|\omega) \frac{\partial E(x;\omega)}{\partial \omega}$ by $\sum_{y,x} P_0(y) K_\omega^m(y, x) \frac{\partial E(x;\omega)}{\partial \omega}$, see equation (5). We use the eigenvector expansion of the transition kernel, equation (6), to express this as $\sum_{y,x,\mu} P_0(y) \{\lambda_\omega^\mu\}^m u_\omega^\mu(y) v_\omega^\mu(x) \frac{\partial E(x;\omega)}{\partial \omega}$. The result follows using the specific forms of the first eigenvectors, see equation (9).*

Result 1 demonstrates that the expected update of CD is similar to the steepest descent rule, see equations (1,10), but with an additional term $\sum_{\mu=2} \{\lambda_\omega^\mu\}^m \{\sum_y P_0(y) u_\omega^\mu(y)\}$ $\{\sum_x v_\omega^\mu(x) \frac{\partial E(x;\omega)}{\partial \omega}\}$ which will be small provided the magnitudes of the eigenvalues $\{\lambda_\omega^\mu\}$ are small for $\mu \geq 2$ (or if the transition kernel can be chosen so that $\sum_y P_0(y) u_\omega^\mu$ is small for $\mu \geq 2$).

We now give a second form for the expected update rule. To do this, we define a new variable $g(x;\omega)$. This is chosen so that $\sum_x P(x|\omega) g(x;\omega) = 0$, $\forall \omega$ and the extrema of the Kullback-Leibler divergence occurs when $\sum_x P_0(x) g(x;\omega) = 0$.

**Result 2.** *Let $g(x;\omega) = \frac{\partial E(x;\omega)}{\partial \omega} - \sum_x P(x|\omega) \frac{\partial E(x;\omega)}{\partial \omega}$, then $\sum_x P(x|\omega) g(x;\omega) = 0$, the extrema of the Kullback-Leibler divergence occur when $\sum_x P_0(x) g(x;\omega) = 0$, and the expected update rule can be written as:*

$$\omega_{t+1} = \omega_t - \gamma_t \{\sum_x P_0(x) g(x;\omega) - \sum_{y,x} P_0(y) K_\omega^m(y, x) g(x;\omega)\}. \tag{11}$$

Proof. *The first result follows directly. The second follows because $\sum_x P_0(x) g(x;\omega) = \sum_x P_0(x) \frac{\partial E(x;\omega)}{\partial \omega} - \sum_x P(x|\omega) \frac{\partial E(x;\omega)}{\partial \omega}$. To get the third we substitute the definition of $g(x;\omega)$ into the expected update equation (5). The result follows using the standard property of transition kernels that $\sum_y K_\omega^m(x, y) = 1$, $\forall x$.*

We now use Results 1 and 2 to understand the fixed points of the CD algorithm and determine whether it is biased.

**Result 3.** *The fixed points $\omega^*$ of the CD algorithm are true (unbiased) extrema of the KL divergence (i.e. $\sum_x P_0(x) g(x;\omega^*) = 0$) if, and only if, we also have $\sum_{y,x} P_0(y) K_{\omega^*}^m(y, x) g(x;\omega^*) = 0$. A sufficient condition is that $P_0(y)$ and $g(x;\omega)$ lie*

*in orthogonal eigenspaces of $K_{\omega^*}(y, x)$. This includes the (known) special case when there exists $\omega^*$ such that $P(x|\omega^*) = P_0(x)$ (see [2]).*

Proof. *The first part follows directly from equation (11) in Result 2. The second part can be obtained by the eigenspace analysis in Result 1. Suppose $P_0(x) = P(x|\omega^*)$. Recall that $v_{\omega^*}^1(x) = P(x|\omega^*)$, and so $\sum_y P_0(y)u_{\omega^{ast}}^\mu(y) = 0$, $\mu \neq 1$. Moreover, $\sum_x v_{\omega^*}^1 g(x; \omega^*) = 0$. Hence $P_0(x)$ and $g(x; \omega^*)$ lie in orthogonal eigenspaces of $K_{\omega^*}(y, x)$.*

Result 3 shows that whether CD converges to an unbiased estimate usually depends on the specific form of the MCMC transition matrix $K_\omega(y, x)$. But there is an intuitive argument why the bias term $\sum_{y,x} P_0(y)K_{\omega^*}^m(y, x)g(x; \omega^*)$ may tend to be small at places where $\sum_x P_0(x)g(x; \omega^*) = 0$. This is because for small $m$, $\sum_y P_0(y)K_{\omega^*}^m(y, x) \approx P_0(x)$ which satisfies $\sum_x P_0(x)g(x; \omega^*) = 0$. Moreover, for large $m$, $\sum_y P_0(y)K_{\omega^*}^m(y, x) \approx P(x|\omega^*)$ and we also have $\sum_x P(x|\omega^*)g(x; \omega^*) = 0$.

Alternatively, using Result 1, the bias term $\sum_{y,x} P_0(y)K_{\omega^*}^m(y, x)g(x; \omega^*)$ can be expressed as $\sum_{\mu=2}\{\lambda_{\omega^*}^\mu\}^m\{\sum_y P_0(y)u_{\omega^*}^\mu(y)\}\{\sum_x v_{\omega^*}^\mu(x)\frac{\partial E(x;\omega^*)}{\partial \omega}\}$. This will tend to be small provided the eigenvalue moduli $|\lambda_{\omega^*}^\mu|$ are small for $\mu \geq 2$ (i.e. the standard conditions for a well defined Markov Chain). In general the bias term should decrease exponentially as $|\lambda_{\omega^*}^2|^m$. Clearly it is also desirable to define the transition kernels $K_\omega(x, y)$ so that the right eigenvectors $\{u_\omega^\mu(y) : \mu \geq 2\}$ are as orthogonal as possible to the observed data $P_0(y)$.

The practicality of CD depends on whether we can find an MCMC sampler such that the bias term $\sum_{y,x} P_0(y)K_{\omega^*}^m(y, x)g(x; \omega^*) = 0$ is small for most $\omega$. If not, then the alternative stochastic algorithms may be preferable.

Finally we give convergence conditions for the CD algorithm.

**Result 4** *CD will converge with probability 1 to state $\omega^*$ provided $\gamma_t = 1/t$, $\frac{\partial E}{\partial \omega}$ is bounded, and*

$$(\omega - \omega^*) \cdot \{\sum_x P_0(x)g(x; \omega) - \sum_{y,x} P_0(y)K_\omega^m(y, x)g(x; \omega)\} \geq K_1|\omega - \omega^*|^2, \quad (12)$$

*for some $K_1$.*

Proof. *This follows from the SAC theorem and Result 2. The boundedness of $\frac{\partial E}{\partial \omega}$ is required to ensure that the "update noise" is bounded in order to satisfy the second condition of the SAC theorem.*

Results 3 and 4 can be combined to ensure that CD converges (with probability 1) to the correct (unbiased) solution. This requires specifying that $\omega^*$ in Result 4 also satisfies the conditions $\sum_x P_0(x)g(x; \omega^*) = 0$ and $\sum_{y,x} P_0(y)K_{\omega^*}^m(y, x)g(x; \omega^*) = 0$.

# 5 Conclusion

The goal of this paper was to relate the Contrastive Divergence (CD) algorithm to the stochastic approximation literature. This enables us to give convergence conditions which ensure that CD will converge to the parameters $\omega^*$ that minimize the Kullback-Leibler divergence $D(P_0(x)||P(x|\omega))$. The analysis also gives necessary and sufficient conditions to determine whether the solution is unbiased. For more recent results, see Carreira-Perpignan and Hinton (in preparation).

The results in this paper are elementary and preliminary. We conjecture that far more

powerful results can be obtained by adapting the convergence theorems in the literature [6,7,9]. In particular, Younes [7] gives convergence results when the gradient of the energy $\partial E(x; \omega)/\partial \omega$ is bounded by a term that is linear in $\omega$ (and hence unbounded). He is also able to analyze the asymptotic behaviour of these algorithms. But adapting his mathematical techniques to Contrastive Divergence is beyond the scope of this paper.

Finally, the analysis in this paper does not seem to capture many of the intuitions behind Contrastive Divergence [1]. But we hope that the techniques described in this paper may also stimulate research in this direction.

### Acknowledgements

I thank Geoff Hinton, Max Welling and Yingnian Wu for stimulating conversations and feedback. Yingnian provided guidance to the stochastic approximation literature and Max gave useful comments on an early draft. This work was partially supported by an NSF SLC catalyst grant "Perceptual Learning and Brain Plasticity" NSF SBE-0350356.

## References

[1]. G. Hinton. "Training Products of Experts by Minimizing Contrastive Divergence"". *Neural Computation*. 14, pp 1771-1800. 2002.

[2]. Y.W. Teh, M. Welling, S. Osindero and G.E. Hinton. "Energy-Based Models for Sparse Overcomplete Representations". *Journal of Machine Learning Research*. To appear. 2003.

[3]. D. MacKay. "Failures of the one-step learning algorithm". Available electronically at http://www.inference.phy.cam.ac.uk/mackay/abstracts/gbm.html. 2001.

[4]. C.K.I. Williams and F.V. Agakov. "An Analysis of Contrastive Divergence Learning in Gaussian Boltzmann Machines". Technical Report EDI-INF-RR-0120. Institute for Adaptive and Neural Computation. University of Edinburgh. 2002.

[5]. H. Robbins and S. Monro. "A Stochastic Approximation Method". *Annals of Mathematical Sciences*. Vol. 22, pp 400-407. 1951.

[6]. H.J. Kushner and D.S. Clark. **Stochastic Approximation for Constrained and Unconstrained Systems.** New York. Springer-Verlag. 1978.

[7]. L. Younes. "On the Convergence of Markovian Stochastic Algorithms with Rapidly Decreasing Ergodicity rates." *Stochastics and Stochastic Reports*, 65, 177-228. 1999.

[8]. S.C. Zhu and X. Liu. "Learning in Gibbsian Fields: How Accurate and How Fast Can It Be?". *IEEE Trans. Pattern Analysis and Machine Intelligence*. Vol. 24, No. 7, July 2002.

[9]. H.J. Kushner. "Asymptotic Global Behaviour for Stochastic Approximation and Diffusions with Slowly Decreasing Noise Effects: Global Minimization via Monte Carlo". *SIAM J. Appl. Math.* 47:169-185. 1987.

[10]. G.B. Orr and T.K. Leen. "Weight Space Probability Densities on Stochastic Learning: II Transients and Basin Hopping Times". *Advances in Neural Information Processing Systems*, 5. Eds. Giles, Hanson, and Cowan. Morgan Kaufmann, San Mateo, CA. 1993.

[11]. G.R. Grimmett and D. Stirzaker. **Probability and Random Processes**. Oxford University Press. 2001.

[12]. B. Van Roy. Course notes. Prof. B. Van Roy. Stanford. (www.stanford.edu/class/msande339/notes/lecture6.ps).

[13]. P. Bremaud. **Markov Chains: Gibbs Fields, Monte Carlo Simulation, and Queues.** Springer. New York. 1999.
